# An Environment Model for Nonstationary Reinforcement Learning

Samuel P. M. Choi        Dit-Yan Yeung        Nevin L. Zhang
pmchoi@cs.ust.hk        dyyeung@cs.ust.hk        lzhang@cs.ust.hk
Department of Computer Science, Hong Kong University of Science and Technology
Clear Water Bay, Kowloon, Hong Kong

## Abstract

Reinforcement learning in nonstationary environments is generally regarded as an important and yet difficult problem. This paper partially addresses the problem by formalizing a subclass of nonstationary environments. The environment model, called *hidden-mode Markov decision process* (HM-MDP), assumes that environmental changes are always confined to a small number of hidden *modes*. A mode basically indexes a Markov decision process (MDP) and evolves with time according to a Markov chain. While HM-MDP is a special case of *partially observable Markov decision processes* (POMDP), modeling an HM-MDP environment via the more general POMDP model unnecessarily increases the problem complexity. A variant of the Baum-Welch algorithm is developed for model learning requiring less data and time.

## 1 Introduction

Reinforcement Learning (RL) [7] is a learning paradigm based upon the framework of Markov decision process (MDP). Traditional RL research assumes that environment dynamics (i.e., MDP parameters) are always fixed (i.e., *stationary*). This assumption, however, is not realistic in many real-world applications. In elevator control [3], for instance, the passenger arrival and departure rates can vary significantly over one day, and should not be modeled by a fixed MDP.

Nonetheless, RL in nonstationary environments is regarded as a difficult problem. In fact, it is an impossible task if there is no regularity in the ways environment dynamics change. Hence, some degree of regularity must be assumed. Typically, nonstationary environments are presummed to change slowly enough such that online RL algorithms can be employed to keep track the changes. The online approach is memoryless in the sense that even if the environment ever revert to the previously learned dynamics, learning must still need to be started all over again.

## 1.1 Our Proposed Model

This paper proposes a formal model [1] for the nonstationary environments that repeats their dynamics in certain ways. Our model is inspired by the observations from the real-world nonstationary tasks with the following properties:

**Property 1.** Environmental changes are confined to a small number of *modes*, which are stationary environments with distinct dynamics. The environment is in exactly one of these modes at any given time. This concept of modes seems to be applicable to many real-world tasks. In an elevator control problem, for example, the system might operate in a morning-rush-hour mode, an evening-rush-hour mode and a non-rush-hour mode. One can also imagine similar modes for other control tasks, such as traffic control and dynamic channel allocation [6].

**Property 2.** Unlike states, modes cannot be directly observed; the current mode can only be estimated according to the past state transitions. It is analogous to the elevator control example in that the passenger arrival rate and pattern can only be inferred through the occurrence of pick-up and drop-off requests.

**Property 3.** Mode transitions are stochastic events and are independent of the control system's responses. In the elevator control problem, for instance, the events that change the current mode of the environment could be an emergency meeting in the administrative office, or a tea break for the staff on the 10th floor. Obviously, the elevator's response has no control over the occurrence of these events.

**Property 4.** Mode transitions are relatively infrequent. In other words, a mode is more likely to retain for some time before switching to another one. If we consider the emergency meeting example, employees on different floors take time to arrive at the administrative office, and thus would generate a similar traffic pattern (drop-off requests on the same floor) for some period of time.

**Property 5.** The number of states is often substantially larger than the number of modes. This is a common property for many real-world applications. In the elevator example, the state space comprises all possible combinations of elevator positions, pick-up and drop-off requests, and certainly would be huge. On the other hand, the mode space could be small. For instance, an elevator control system can simply have the three modes as described above to approximate the reality.

Based on these properties, an environment model is proposed by introducing a mode variable to capture environmental changes. Each mode specifies an MDP and hence completely determines the current state transition function and reward function (property 1). A mode, however, is not directly observable (property 2), and evolves with time according to a Markov process (property 3). The model is therefore called *hidden-mode model*. Note that our model does not impose any constraint to satisfy properties 4 and 5. In other words, the hidden-mode model can work for environments without these two properties. Nevertheless, as will be shown later, these properties can improve learning in practice.

## 1.2 Related Work

Our hidden-mode model is related to a nonstationary model proposed by Dayan and Sejnowski [4]. Although our model is more restrictive in terms of representational power, it involves much fewer parameters and is thus easier to learn. Besides, other than the number of possible modes, we do not assume any other knowledge about

the way environment dynamics change. Dayan and Sejnowski, on the other hand, assume that one knows precisely how the environment dynamics change.

The hidden-mode model can also be viewed as a special case of the hidden-state model, or *partially observable Markov decision process* (POMDP). As will be shown later, a hidden-mode model can always be represented by a hidden-state model through state augmentation. Nevertheless, modeling a hidden-mode environment via a hidden-state model will unnecessarily increase the problem complexity. In this paper, the conversion from the former to the latter is also briefly discussed.

## 1.3 Our Focus

There are two approaches for RL. Model-based RL first acquires an environment model and then, from which, an optimal policy is derived. Model-free RL, on the contrary, learns an optimal policy directly through its interaction with the environment. This paper is concerned with the first part of the model-based approach, i.e., how a hidden-mode model can be learned from experience. We will address the policy learning problem in a separate paper.

## 2   Hidden-Mode Markov Decision Processes

This section presents our hidden-mode model. Basically, a hidden-mode model is defined as a finite set of MDPs that share the same state space and action space, with possibly different transition functions and reward functions. The MDPs correspond to different modes in which a system operates. States are completely observable and their transitions are governed by an MDP. In contrast, modes are not directly observable and their transitions are controlled by a Markov chain. We refer to such a process as a *hidden-mode Markov decision process* (HM-MDP). An example of HM-MDP is shown in Figure 1(a).

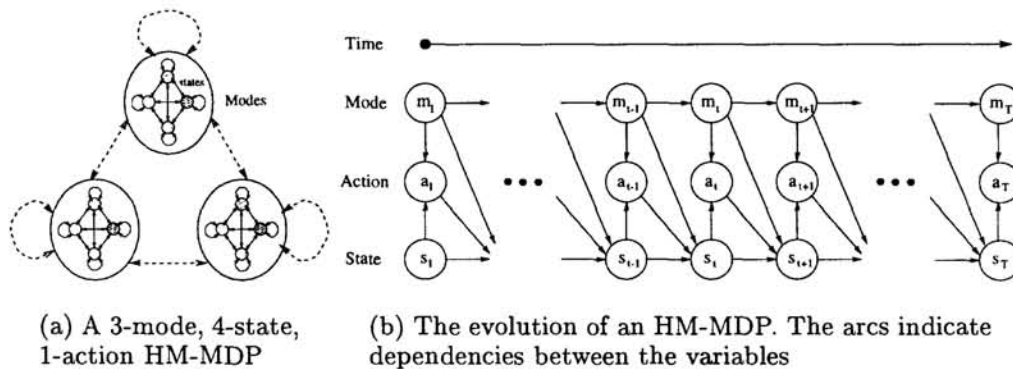

(a) A 3-mode, 4-state, 1-action HM-MDP

(b) The evolution of an HM-MDP. The arcs indicate dependencies between the variables

Figure 1: An HM-MDP

Formally, an HM-MDP is an 8-tuple $(Q, S, A, X, Y, R, \Pi, \Psi)$, where $Q$, $S$ and $A$ represent the sets of modes, states and actions respectively; the mode transition function $X$ maps mode $m$ to $n$ with a fixed probability $x_{mn}$; the state transition function $Y$ defines transition probability, $y_m(s, a, s')$, from state $s$ to $s'$ given mode $m$ and action $a$; the stochastic reward function $R$ returns rewards with mean value $r_m(s, a)$; $\Pi$ and $\Psi$ denote the prior probabilities of the modes and of the states respectively. The evolution of modes and states over time is depicted in Figure 1(b).

HM-MDP is a subclass of POMDP. In other words, the former can be reformulated as a special case of the latter. Specifically, one may take an ordered pair of any mode and observable state in the HM-MDP as a hidden state in the POMDP, and any observable state of the former as an observation of the latter. Suppose the observable states $s$ and $s'$ are in modes $m$ and $n$ respectively. These two HM-MDP states together with their corresponding modes form two hidden states $(m, s)$ and $(n, s')$ for its POMDP counterpart. The transition probability from $(m, s)$ to $(n, s')$ is then simply the mode transition probability $x_{mn}$ multiplied by the state transition probability $y_m(s, a, s')$. For an $M$-mode, $N$-state, $K$-action HM-MDP, the equivalent POMDP thus has $N$ observations and $MN$ hidden states. Since most state transition probabilities are collapsed into mode transition probabilities through parameter sharing, the number of parameters in an HM-MDP $(N^2 MK + M^2)$ is much less than that of its corresponding POMDP $(M^2 N^2 K)$.

## 3  Learning a Hidden-Mode Model

There are now two ways to learn a hidden-mode model. One may learn either an HM-MDP, or an equivalent POMDP instead. POMDP models can be learned via a variant of the Baum-Welch algorithm [2]. This POMDP Baum-Welch algorithm requires $\Theta(M^2 N^2 T)$ time and $\Theta(M^2 N^2 K)$ storage for learning an $M$-mode, $N$-state, $K$-action HM-MDP, given $T$ data items.

A similar idea can be applied to the learning of an HM-MDP. Intuitively, one can estimate the model parameters based on the expected counts of the mode transitions, computed by a set of auxiliary variables. The major difference from the original algorithm is that consecutive state transitions, rather than the observations, are considered. Additional effort is thus needed for handling the boundary cases. This HM-MDP Baum-Welch algorithm is described in Figure 2.

## 4  Empirical Studies

This section empirically examines the POMDP Baum-Welch[1] and HM-MDP Baum-Welch algorithms. Experiments based on various randomly generated models and some real-world environments were conducted. The results are quite consistent. For illustration, a simple traffic control problem is presented. In this problem, one direction of a two-way traffic is blocked, and cars from two different directions (left and right) are forced to share the remaining road. To coordinate the traffic, two traffic lights equipped with sensors are set. The system then has two possible actions: either to signal cars from the left or cars from the right to pass. For simplicity, we assume discrete time steps and uniform speed of the cars.

The system has 8 possible states; they correspond to the combinations of whether there are cars waiting on the left and the right directions, and the stop signal position in the previous time step. There are 3 traffic modes. The first one has cars waiting on the left and the right directions with probabilities 0.3 and 0.1 respectively. In the second mode, these probabilities are reversed. For the last one, both probabilities are 0.3. In addition, the mode transition probability is 0.1. A cost of -1.0 results if

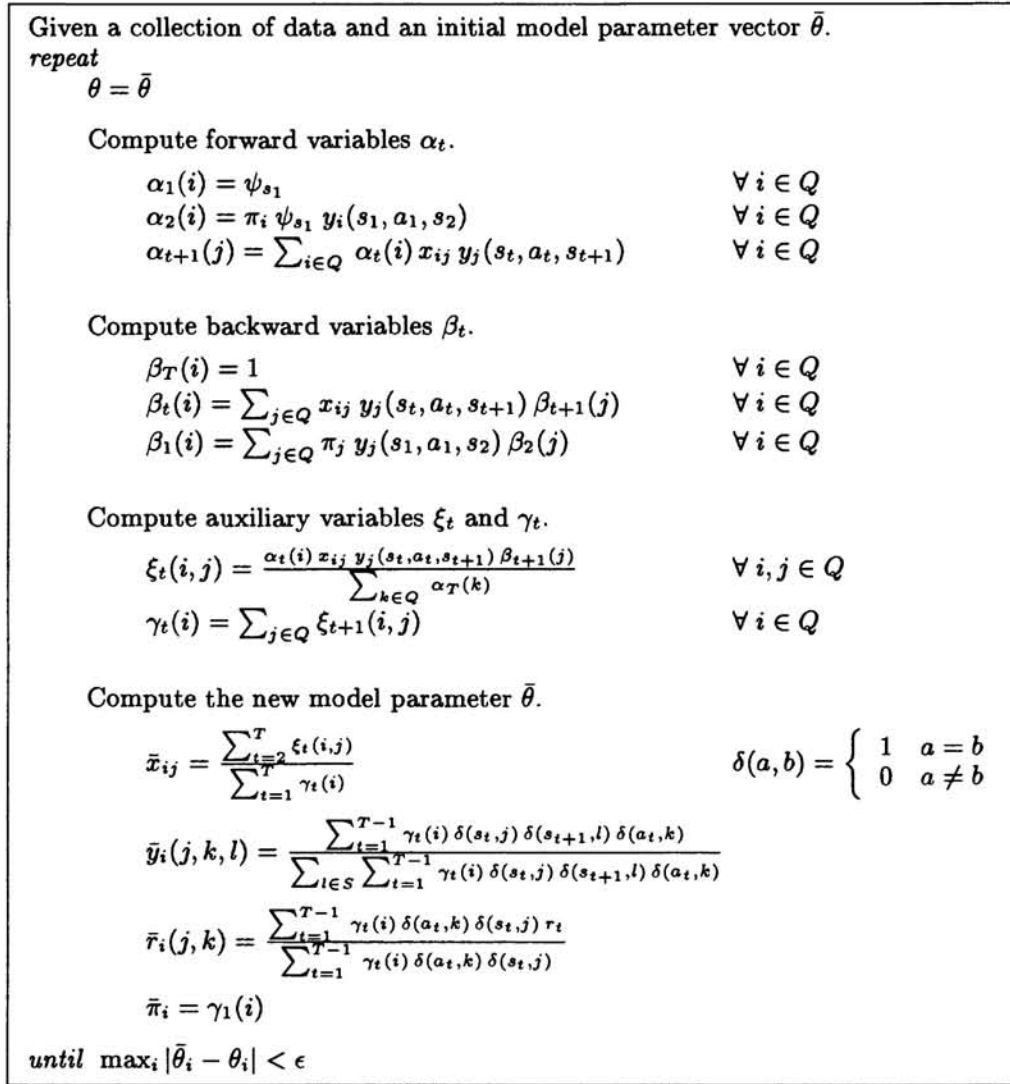

Given a collection of data and an initial model parameter vector $\bar{\theta}$.

*repeat*

$\quad \theta = \bar{\theta}$

Compute forward variables $\alpha_t$.

$$\alpha_1(i) = \psi_{s_1} \qquad\qquad \forall\, i \in Q$$
$$\alpha_2(i) = \pi_i\, \psi_{s_1}\, y_i(s_1, a_1, s_2) \qquad\qquad \forall\, i \in Q$$
$$\alpha_{t+1}(j) = \sum_{i \in Q} \alpha_t(i)\, x_{ij}\, y_j(s_t, a_t, s_{t+1}) \qquad\qquad \forall\, i \in Q$$

Compute backward variables $\beta_t$.

$$\beta_T(i) = 1 \qquad\qquad \forall\, i \in Q$$
$$\beta_t(i) = \sum_{j \in Q} x_{ij}\, y_j(s_t, a_t, s_{t+1})\, \beta_{t+1}(j) \qquad\qquad \forall\, i \in Q$$
$$\beta_1(i) = \sum_{j \in Q} \pi_j\, y_j(s_1, a_1, s_2)\, \beta_2(j) \qquad\qquad \forall\, i \in Q$$

Compute auxiliary variables $\xi_t$ and $\gamma_t$.

$$\xi_t(i,j) = \frac{\alpha_t(i)\, x_{ij}\, y_j(s_t, a_t, s_{t+1})\, \beta_{t+1}(j)}{\sum_{k \in Q} \alpha_T(k)} \qquad\qquad \forall\, i, j \in Q$$
$$\gamma_t(i) = \sum_{j \in Q} \xi_{t+1}(i,j) \qquad\qquad \forall\, i \in Q$$

Compute the new model parameter $\bar{\theta}$.

$$\bar{x}_{ij} = \frac{\sum_{t=2}^{T} \xi_t(i,j)}{\sum_{t=1}^{T} \gamma_t(i)} \qquad\qquad \delta(a,b) = \left\{ \begin{array}{ll} 1 & a = b \\ 0 & a \neq b \end{array} \right.$$

$$\bar{y}_i(j,k,l) = \frac{\sum_{t=1}^{T-1} \gamma_t(i)\, \delta(s_t, j)\, \delta(s_{t+1}, l)\, \delta(a_t, k)}{\sum_{l \in S} \sum_{t=1}^{T-1} \gamma_t(i)\, \delta(s_t, j)\, \delta(s_{t+1}, l)\, \delta(a_t, k)}$$

$$\bar{r}_i(j,k) = \frac{\sum_{t=1}^{T-1} \gamma_t(i)\, \delta(a_t, k)\, \delta(s_t, j)\, r_t}{\sum_{t=1}^{T-1} \gamma_t(i)\, \delta(a_t, k)\, \delta(s_t, j)}$$

$$\bar{\pi}_i = \gamma_1(i)$$

*until* $\max_i |\bar{\theta}_i - \theta_i| < \epsilon$

Figure 2: HM-MDP Baum-Welch Algorithm

a car waits on either side.

The experiments were run with the same initial model for data sets of various sizes. The algorithms iterated until the maximum change of the model parameters was less than a threshold of 0.0001. The experiment was repeated for 20 times with different random seeds in order to compute the median. Then the learned models were compared in their POMDP forms using the Kullback-Leibler (KL) distance [5], and the total CPU running time on a SUN Ultra I workstation was measured. Figure 3 (a) and (b) report the results.

Generally speaking, both algorithms learn a more accurate environment model as the data size increases (Figure 3 (a)). This result is expected as both algorithms are statistically-based, and hence their performance relies largely on the data size. When the training data size is very small, both algorithms perform poorly. However, as the data size increases, HM-MDP Baum-Welch improves substantially faster than POMDP Baum-Welch. It is because an HM-MDP in general consists of fewer free

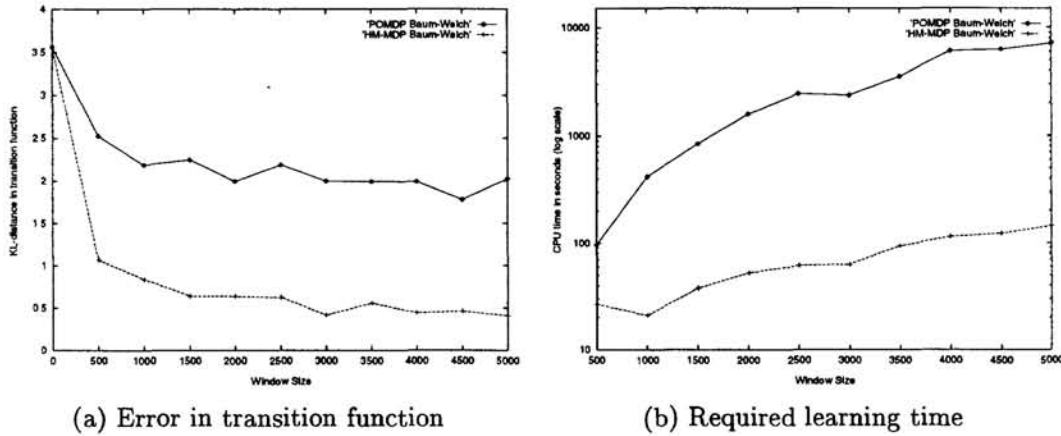

(a) Error in transition function              (b) Required learning time

Figure 3: Empirical results on model learning

parameters than its POMDP counterpart.

HM-MDP Baum-Welch also runs much faster than POMDP Baum-Welch (Figure 3 (b)). It holds in general for the same reason discussed above. Note that computational time is not necessarily monotonically increasing with the data size. It is because the total computation depends not only on the data size, but also on the number of iterations executed. From our experiments, we noticed that the number of iterations tends to decrease as the data size increases.

Larger models have also been tested. While HM-MDP Baum-Welch is able to learn models with several hundred states and a few modes, POMDP Baum-Welch was unable to complete the learning in a reasonable time. Additional experimental results can be found in [1].

## 5    Discussions and Future Work

The usefulness of a model depends on the validity of the assumptions made. We now discuss the assumptions of HM-MDP, and shed some light on its applicability to real-world nonstationary tasks. Some possible extensions are also discussed.

**Modeling a nonstationary environment as a number of distinct MDPs.** MDP is a flexible framework that has been widely adopted in various applications. Modeling nonstationary environments by distinct MDPs is a natural extension to those tasks. Comparing to POMDP, our model is more comprehensive: each MDP naturally describes a mode of the environment. Moreover, this formulation facilitates the incorporation of prior knowledge into the model initialization step.

**States are directly observable while modes are not.** While completely observable states are helpful to infer the current mode, it is also possible to extend the model to allow partially observable states. In this case, the extended model would be equivalent in representational power to a POMDP. This could be proved easily by showing the reformulation of the two models in both directions.

**Mode changes are independent of the agent's responses.** This property may not always hold for all real-world tasks. In some applications, the agent's actions might affect the state as well as the environment mode. In that case, an MDP should be used to govern the mode transition process.

**Mode transitions are relatively infrequent.** This is a property that generally holds in many applications. Our model, however, is not limited by this condition. We have tried to apply our model-learning algorithms to problems in which this property does not hold. We find that our model still outperforms POMDP, although the required data size is typically larger for both models.

**Number of states is substantially larger than the number of modes.** This is the key property that significantly reduces the number of parameters in HM-MDP compared to that in POMDP. In practice, introduction of a few modes is sufficient for boosting the system performance. More modes might only help little. Thus a trade-off between performance and response time must be decided.

There are additional issues that need to be addressed. First, an efficient algorithm for policy learning is required. Although in principle it can be achieved indirectly via any POMDP algorithm, a more efficient algorithm based on the model-based approach is possible. We will address this issue in a separate paper. Next, the number of modes is currently assumed to be known. We are now investigating how to remove this limitation. Finally, the exploration-exploitation issue is currently ignored. In our future work, we will address this important issue and apply our model to real-world nonstationary tasks.

## Footnotes

[1]Chrisman's algorithm also attempts to learn a minimal possible number of states. Our paper concerns only with learning the model parameters.

# References

[1] S. P. M. Choi, D. Y. Yeung, and N. L. Zhang. Hidden-mode Markov decision processes. In *IJCAI 99 Workshop on Neural, Symbolic, and Reinforcement Methods for Sequence Learning*, 1999.

[2] L. Chrisman. Reinforcement learning with perceptual aliasing: The perceptual distinctions approach. In *AAAI-92*, 1992.

[3] R. H. Crites and A. G. Barto. Improving elevator performance using reinforcement learning. In D. Touretzky, M. Mozer, and M. Hasselmo, editors, *Advances in Neural Information Processing Systems 8*, 1996.

[4] P. Dayan and T. J. Sejnowski. Exploration bonuses and dual control. *Machine Learning*, 25(1):5–22, Oct. 1996.

[5] S. Kullback. *Information Theory and Statistics*. Wiley, New York, NY, USA, 1959.

[6] S. Singh and D. P. Bertsekas. Reinforcement learning for dynamic channel allocation in cellular telephone systems. In *Advances in Neural Information Processing Systems 9*, 1997.

[7] R. S. Sutton and A. G. Barto. *Reinforcement Learning: An Introduction*. The MIT Press, 1998.
